# Particle Filter-based Policy Gradient in POMDPs

**Pierre-Arnaud Coquelin**
CMAP, Ecole Polytechnique
coquelin@cmapx.polytechnique.fr

**Romain Deguest**[*]
CMAP, Ecole Polytechnique
deguest@cmapx.polytechnique.fr

**Rémi Munos**
INRIA Lille - Nord Europe, SequeL project,
remi.munos@inria.fr

## Abstract

Our setting is a Partially Observable Markov Decision Process with continuous state, observation and action spaces. Decisions are based on a Particle Filter for estimating the belief state given past observations. We consider a policy gradient approach for parameterized policy optimization. For that purpose, we investigate sensitivity analysis of the performance measure with respect to the parameters of the policy, focusing on Finite Difference (FD) techniques. We show that the naive FD is subject to variance explosion because of the non-smoothness of the resampling procedure. We propose a more sophisticated FD method which overcomes this problem and establish its consistency.

## 1 Introduction

We consider a Partially Observable Markov Decision Problem (POMDP) (see e.g. (Lovejoy, 1991; Kaelbling et al., 1998)) defined by a state process $(X_t)_{t\geq 1} \in X$, an observation process $(Y_t)_{t\geq 1} \in Y$, a decision (or action) process $(A_t)_{t\geq 1} \in A$ which depends on a policy (mapping from all possible observation histories to actions), and a reward function $r : X \to \mathbb{R}$. Our goal is to find a policy $\pi$ that maximizes a performance measure $J(\pi)$, function of future rewards, for example in a finite horizon setting:

$$J(\pi) \stackrel{\text{def}}{=} \mathbb{E}\Big[\sum_{t=1}^{n} r(X_t)\Big]. \tag{1}$$

Other performance measures (such as in infinite horizon with discounted rewards) could be handled as well. In this paper, we consider the case of **continuous state, observation, and action spaces**.

The **state process** is a Markov decision process taking its values in a (measurable) state space $X$, with initial probability measure $\mu \in \mathcal{M}(X)$ (i.e. $X_1 \sim \mu$), and which can be simulated using a transition function $F$ and independent random numbers, i.e. for all $t \geq 1$,

$$X_{t+1} = F(X_t, A_t, U_t), \text{ with } U_t \stackrel{i.i.d.}{\sim} \nu, \tag{2}$$

where $F : X \times A \times U \to X$ and $(U, \sigma(U), \nu)$ is a probability space. In many practical situations $U = [0,1]^p$ and $U_t$ is a $p$-uple of pseudo random numbers. For simplicity, we adopt the notations $F(x_0, a_0, u) \stackrel{\text{def}}{=} F_\mu(u)$, where $F_\mu$ is the first transition function (i.e. $X_1 = F_\mu(U_0)$ with $U_0 \sim \nu$).

The **observation process** $(Y_t)_{t\geq 1}$ lies in a (measurable) space $Y$ and is linked with the state process by the conditional probability measure $\mathbb{P}(Y_t \in dy_t | X_t = x_t) = g(x_t, y_t)\, dy_t$, where $g : X \times Y \to [0,1]$ is the marginal density function of $Y_t$ given $X_t$. We assume that observations are conditionally independent given the state process. Here also, we assume that we can simulate an observation using a transition function $G$ and independent random numbers, i.e. $\forall t \geq 1$, $Y_t = G(X_t, V_t)$,

---

[*]Also affiliated to Columbia University

where $V_t \overset{i.i.d.}{\sim} \nu$ (for the sake of simplicity we consider the same probability space $(U, \sigma(U), \nu)$). Now, the **action process** $(A_t)_{t \geq 1}$ depends on a **policy** $\pi$ which assigns to each possible observation history $Y_{1:t}$ (where we adopt the usual notation "$1:t$" to denote the collection of integers $s$ such that $1 \leq s \leq t$), an action $A_t \in A$.

In this paper we will consider policies that depend on the **belief state** (also called **filtering distribution**) conditionally to past observations. The belief state, written $b_t$, belongs to $\mathcal{M}(X)$ (the space of all probability measures on $X$) and is defined by $b_t(dx_t, Y_{1:t}) \overset{\text{def}}{=} \mathbb{P}(X_t \in dx_t | Y_{1:t})$, and will be written $b_t(dx_t)$ or even $b_t$ for simplicity when there is no risk of confusion. Because of the Markov property of the state dynamics, the belief state $b_t(\cdot, Y_{1:t})$ is the most informative representation about the current state $X_t$ given the history of past observations $Y_{1:t}$. It represents sufficient statistics for designing an optimal policy in the class of observations-based policies.

The temporal and causal dependencies of the dynamics of a generic POMDP using belief-based policies is summarized in Figure 1 (left): at time $t$, the state $X_t$ is unknown, only $Y_t$ is observed, which enables (at least in theory) to update $b_t$ based on the previous belief $b_{t-1}$. The policy $\pi$ takes as input the belief state $b_t$ and returns an action $A_t$ (the policy may be deterministic or stochastic). However, since the belief state is an infinite dimensional object, and thus cannot be represented in a computer, we first simplify the class of policies that we consider here to be defined over a finite dimensional space of **belief-features** $f : \mathcal{M}(X) \to \mathbb{R}^K$ which represents relevant statistics of the filtering distribution. We write $b_t(f_k)$ for the value of the $k$-th feature (among $K$) (where we use the usual notation $b(f) \overset{\text{def}}{=} \int_X f(x)b(dx)$ for any function $f$ defined on $X$ and measure $b \in \mathcal{M}(X)$), and denote $b_t(f)$ the vector (of size $K$) with components $b_t(f_k)$. Examples of features are: $f(x) = x$ (mean value), $f(x) = x'x$ (for the covariance matrix). Other more complex features (e.g. entropy measure) could be used as well. Such a policy $\pi : \mathbb{R}^K \to A$ selects an action $A_t = \pi(b_t(f))$, which in turn, yields a new state $X_{t+1}$.

Except for simple cases, such as in finite-state finite-observation processes (where a Viterbi algorithm could be applied (Rabiner, 1989)), and the case of linear dynamics and Gaussian noise (where a Kalman filter could be used), there is no closed-form representation of the belief state. Thus $b_t$ must be approximated in our general setting. A popular method for approximating the filtering distribution is known as **Particle Filters** (PF) (also called **Interacting Particle Systems** or **Sequential Monte-Carlo**). Such particle-based approaches have been used in many applications (see e.g. (Doucet et al., 2001) and (Del Moral, 2004) for a Feynman-Kac framework) for example for parameter estimation in Hidden Markov Models and control (Andrieu et al., 2004) and mobile robot localization (Fox et al., 2001). An PF approximates the belief state $b_t \in \mathcal{M}(X)$ by a set of particles $(x_t^{1:N})$ (points of $X$), which are updated sequentially at each new observation by a transition-selection procedure. In particular, the belief feature $b_t(f)$ is approximated by $\frac{1}{N}\sum_{i=1}^{N} f(x_t^i)$, and the policy is thus a function that takes as input the activation of the feature $f$ at the position of the particles: $A_t = \pi(\frac{1}{N}\sum_{i=1}^{N} f(x_t^i))$. For such methods, the general scheme for POMDPs using Particle Filter-based policies is described in Figure 1 (right).

In this paper, we consider a class of policies $\pi_\theta$ parameterized by a (multi-dimensional) parameter $\theta$ and we search for the value of $\theta$ that maximizes the resulting criterion $J(\pi_\theta)$, now written $J(\theta)$ for simplicity. We focus on a policy gradient approach: the POMDP is replaced by an optimization problem on the space of policy parameters, and a (stochastic) gradient ascent on $J(\theta)$ is considered. For that purpose (and this is the object of this work) we investigate the estimation of $\nabla J(\theta)$ (where the gradient $\nabla$ refers to the derivative w.r.t. $\theta$), with an emphasis on Finite-Difference techniques. There are many works about such policy gradient approach in the field of Reinforcement Learning, see e.g. (Baxter & Bartlett, 1999), but the policies considered are generally not based on the result of an PF. Here, we explicitly consider a class of policies that are based on a belief state constructed by a PF. Our motivations for investigating this case are based on two facts: (1) the belief state represents sufficient statistics for optimality, as mentioned above. (2) PFs are a very popular and efficient tool for constructing the belief state in continuous domains.

After recalling the general approach for evaluating the performance of a PF-based policy (Section 2), we describe (in Section 3.1) a naive Finite-Difference (FD) approach (defined by a step size $h$) for estimating $\nabla J(\theta)$. We discuss the bias and variance tradeoff and explain the problem of variance explosion when $h$ is small. This problem is a consequence of the discontinuity of the resampling operation w.r.t. the parameter $\theta$. Our contribution is detailed in Section 3.2: We propose a modified

FD estimate for $\nabla J(\theta)$ which (along the random sample path) has bias $O(h^2)$ and variance $O(1/N)$, thus overcomes the drawback of the previous naive method. An algorithm is described and illustrated in Section 4 on a simple problem where the optimal policy exhibits a tradeoff between greedy reward optimization and localization.

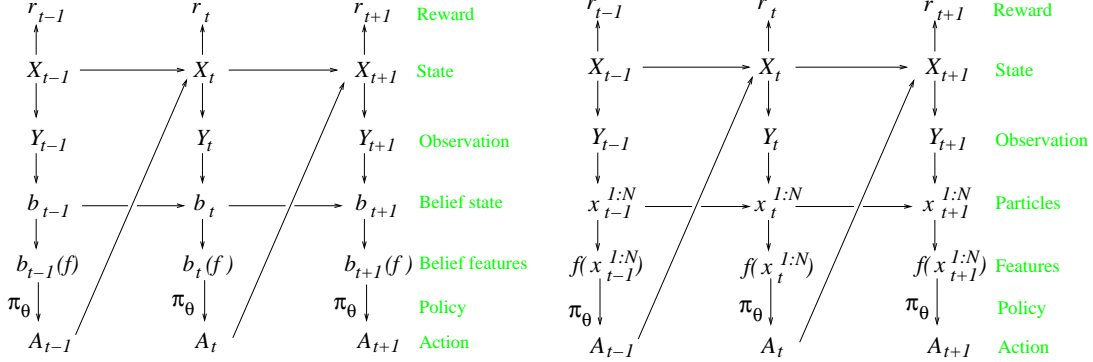

Figure 1: Left figure: Causal and temporal dependencies in a POMDP. Right figure: PF-based scheme for POMDPs where the belief feature $b_t(f)$ is approximated by $\frac{1}{N}\sum_{i=1}^{N} f(x_t^i)$.

## 2 Particle Filters (PF)

We first describe a generic PF for estimating the belief state based on past observations. In Subsection 2.1 we detail how to control a real-world POMDP and in Subsection 2.2 how to estimate the performance of a given policy in simulation. In both cases, we assume that the models of the dynamics (state, observation) are known. The basic PF, called Bootstrap Filter, see (Doucet et al., 2001) for details, approximates the belief state $b_n$ by an empirical distribution $b_n^N \stackrel{\text{def}}{=} \sum_{i=1}^{N} w_n^i \delta_{x_n^i}$ (where $\delta$ denotes a Dirac distribution) made of $N$ particles $x_n^{1:N}$. It consists in iterating the two following steps: at time $t$, given observation $y_t$,

- **Transition step:** (also called **importance sampling** or **mutation**) a successor particles population $\widetilde{x}_t^{1:N}$ is generated according to the state dynamics from the previous population $x_{t-1}^{1:N}$. The (importance sampling) weights $w_t^{1:N} \stackrel{\text{def}}{=} \frac{g(\widetilde{x}_t^{1:N}, y_t)}{\sum_{j=1}^{N} g(\widetilde{x}_t^j, y_t)}$ are evaluated,

- **Selection step:** Resample (with replacement) $N$ particles $x_t^{1:N}$ from the set $\widetilde{x}_t^{1:N}$ according to the weights $w_t^{1:N}$. We write $x_t^{1:N} \stackrel{\text{def}}{=} \widetilde{x}_t^{k_t^{1:N}}$ where $k_t^{1:N}$ are the selection indices.

Resampling is used to avoid the problem of degeneracy of the algorithm, i.e. that most of the weights decreases to zero. It consists in selecting new particle positions such as to preserve a consistency property (i.e. $\sum_{i=1}^{N} w_t^i \phi(\widetilde{x}_t^i) = \mathbb{E}[\frac{1}{N}\sum_{i=1}^{N} \phi(x_t^i)]$). The simplest version introduced in (Gordon et al., 1993) chooses the selection indices $k_t^{1:N}$ by an independent sampling from the set $1:N$ according to a multinomial distribution with parameters $w_t^{1:N}$, i.e. $\mathbb{P}(k_t^i = j) = w_t^j$, for all $1 \leq i \leq N$. The idea is to replicate the particles in proportion to their weights. Many variants have been proposed in the literature, among which the stratified resampling method (Kitagawa, 1996) which is optimal in terms of variance, see e.g. (Cappé et al., 2005).

Convergence issues of $b_n^N(f)$ to $b_n(f)$ (e.g. Law of Large Numbers or Central Limit Theorems) are discussed in (Del Moral, 2004) or (Douc & Moulines, 2008). For our purpose we note that under weak conditions on the feature $f$, we have the consistency property: $b^N(f) \rightarrow b(f)$, almost surely.

### 2.1 Control of a real system by an PF-based policy

We describe in Algorithm 1 how one may use an PF-based policy $\pi_\theta$ for the control of a real-world system. Note that from our definition of $F_\mu$, the particles are initialized with: $\widetilde{x}_1^{1:N} \stackrel{iid}{\sim} \mu$.

### 2.2 Estimation of $J(\theta)$ in simulation

Now, for the purpose of policy optimization, one should be capable of evaluating the performance of a policy in simulation. $J(\theta)$, defined by (1), may be estimated in simulation provided that

---
**Algorithm 1** Control of a real-world POMDP
---
$\quad$ **for** $t = 1$ **to** $n$ **do**
$\quad\quad$ **Observe:** $y_t$,
$\quad\quad$ **Particle transition step:**
$\quad\quad$ Set $\widetilde{x}_t^{1:N} = F(x_{t-1}^{1:N}, a_{t-1}, u_{t-1}^{1:N})$ with $u_{t-1}^{1:N} \overset{iid}{\sim} \nu$. Set $w_t^{1:N} = \frac{g(\widetilde{x}_t^{1:N}, y_t)}{\sum_{j=1}^{N} g(\widetilde{x}_t^{j}, y_t)}$,
$\quad\quad$ **Particle resampling step:**
$\quad\quad$ Set $x_t^{1:N} = \widetilde{x}_t^{k_t^{1:N}}$ where $k_t^{1:N}$ are given by the selection step according to the weights $w_t^{1:N}$.
$\quad\quad$ **Select action:** $a_t = \pi_\theta(\frac{1}{N}\sum_{i=1}^{N} f(x_t^i))$,
$\quad$ **end for**
---

the dynamics of the state and observation are known. Making explicit the dependency w.r.t. the random sample path, written $\omega$ (which accounts for the state and observation stochastic dynamics *and* the random numbers used in the PF-based policy), we write $J(\theta) = \mathbb{E}_\omega[J_\omega(\theta)]$, where $J_\omega(\theta) \overset{\text{def}}{=} \sum_{t=1}^{n} r(X_{t,\omega}(\theta))$, making the dependency of the state w.r.t. $\omega$ and $\theta$ explicit.

Algorithm 2 describes how to evaluate an PF-based policy in simulation. The function returns an estimate, written $J_\omega^N(\theta)$, of $J_\omega(\theta)$. Using previously mentioned asymptotic convergence results for PF, one has $\lim_{N\to\infty} J_\omega^N(\theta) = J_\omega(\theta)$, almost surely (a.s.). In order to approximate $J(\theta)$, one would perform several calls to the algorithm, receiving $J_{\omega_m}^N(\theta)$ (for $1 \le m \le M$), and calculate their empirical mean $\frac{1}{M}\sum_{m=1}^{M} J_{\omega_m}^N(\theta)$, which tends to $J(\theta)$ a.s., when $M, N \to \infty$.

---
**Algorithm 2** Estimation of $J_\omega(\theta)$ in simulation
---
$\quad$ **for** $t = 1$ **to** $n$ **do**
$\quad\quad$ **Define state:**
$\quad\quad$ $x_t = F(x_{t-1}, a_{t-1}, u_{t-1})$ with $u_{t-1} \sim \nu$,
$\quad\quad$ **Define observation:**
$\quad\quad$ $y_t = G(x_t, v_t)$ with $v_t \sim \nu$,
$\quad\quad$ **Particle transition step:**
$\quad\quad$ Set $\widetilde{x}_t^{1:N} = F(x_{t-1}^{1:N}, a_{t-1}, u_{t-1}^{1:N})$ with $u_{t-1}^{1:N} \overset{iid}{\sim} \nu$. Set $w_t^{1:N} = \frac{g(\widetilde{x}_t^{1:N}, y_t)}{\sum_{j=1}^{N} g(\widetilde{x}_t^{j}, y_t)}$,
$\quad\quad$ **Particle resampling step:**
$\quad\quad$ Set $x_t^{1:N} = \widetilde{x}_t^{k_t^{1:N}}$ where $k_t^{1:N}$ are given by the selection step according to the weights $w_t^{1:N}$,
$\quad\quad$ **Select action:** $a_t = \pi_\theta(\frac{1}{N}\sum_{i=1}^{N} f(x_t^i))$,
$\quad$ **end for**
$\quad$ **Return** $J_\omega^N(\theta) \overset{\text{def}}{=} \sum_{t=1}^{n} r(x_t)$.
---

## 3 A policy gradient approach

Now we want to optimize the value of the parameter *in simulation*. Then, once a "good" parameter $\theta^*$ is found, we would use Algorithm 1 to control the real system using the corresponding PF-based policy $\pi_{\theta^*}$. Gradient approaches have been studied in the field of continuous space Hidden Markov Models in (Fichoud et al., 2003; Cérou et al., 2001; Doucet & Tadic, 2003). The authors have used a *likelihood ratio* approach to evaluate $\nabla J(\theta)$. Such methods suffer from high variance, in particular for problems with small noise. In order to reduce the variance, it has been proposed in (Poyadjis et al., 2005) to use a marginal particle filter instead of a simple path-based particle filter. This approach is efficient in terms of variance reduction but its computational complexity is $O(N^2)$.

Here we investigate a pathwise (i.e. along the random sample path $\omega$) sensitivity analysis of $J_\omega(\theta)$ (w.r.t. $\theta$) for the purpose of (stochastic) gradient optimization. We start with a naive Finite Difference (FD) approach and show the problem of variance explosion. Then we provide an alternative, called **common indices FD**, which overcomes this problem.

In the sequel, we make the assumptions that all relevant functions ($F$, $g$, $f$, $\pi$) are continuously differentiable w.r.t. their respective variables. Note that although this is not explicitly mentioned, all such functions may depend on time.

## 3.1 Naive Finite-Difference (FD) method

Let us consider the derivative of $J(\theta)$ component-wisely, writing $\partial J(\theta)$ the derivative of $J(\theta)$ w.r.t. a one-dimensional parameter. If the parameter $\theta$ is multi-dimensional, the derivative will be calculated in each direction. For $h > 0$ we define the centered finite-difference quotient $I_h \stackrel{\text{def}}{=} \frac{J(\theta+h)-J(\theta-h)}{2h}$. Since $J(\theta)$ is differentiable then $\lim_{h\to 0} I_h = \partial J(\theta)$. Consequently, a method for approximating $\partial J(\theta)$ would consist in estimating $I_h$ for a sufficiently small $h$. We know that $J(\theta)$ can be numerically estimated by $\frac{1}{M} \sum_{m=1}^M J_{\omega_m}^N(\theta)$. Thus, it seems natural to estimate $I_h$ by

$$I_h^{N,M} \stackrel{\text{def}}{=} \frac{1}{2h} \Big[ \frac{1}{M} \sum_{m=1}^M J_{\omega_m}^N(\theta+h) - \frac{1}{M} \sum_{m'=1}^M J_{\omega_{m'}}^N(\theta-h) \Big]$$

where we used independent random numbers to evaluate $J(\theta + h)$ and $J(\theta - h)$. From the consistency of the PF, we deduce that $\lim_{h\to 0} \lim_{M,N\to\infty} I_h^{N,M} = \partial J(\theta)$. This naive FD estimate exhibits the following bias-variance tradeoff (see (Coquelin et al., 2008) for the proof):

**Proposition 1** (Bias-variance trade-off). *Assume that $J(\theta)$ is three times continuously differentiable in a small neighborhood of $\theta$, then the asymptotic (when $N \to \infty$) bias of the naive FD estimate $I_h^{N,M}$ is of order $O(h^2)$ and its variance is $O(N^{-1}M^{-1}h^{-2})$.*

In order to reduce the bias, one should choose a small $h$, but then the variance would blow up. Additional computational resource (larger number of particles $N$) will help controlling the variance. However, in practice, e.g. for stochastic optimization, this leads to an intractable amount of computational effort since any consistent FD-based optimization algorithm (e.g. such as the Kiefer-Wolfowitz algorithm) will need to consider a sequence of steps $h$ that decreases with the number of gradient iterations. But if the number of particles is bounded, the variance term will diverge, which may prevent the stochastic gradient algorithm from converging to a local optimum.

In order to reduce the variance of the previous estimator when $h$ is small, one may use *common random numbers* to estimate both $J(\theta + h)$ and $J(\theta - h)$ (i.e. $\omega_m = \omega_{m'}$). The variance then reduces to $O(N^{-1}M^{-1}h^{-1})$ (see e.g. (Glasserman, 2003)), which still explodes for small $h$.

Now, under the additional assumption that along almost all random sample path $\omega$, the function $\theta \mapsto J_\omega^N(\theta)$ is a.s. continuous, then the variance would reduce to $O(N^{-1}M^{-1})$ (see Section (7.1) of (Glasserman, 2003)). Unfortunately, this is not the case here because of the discontinuity of the PF resampling operation w.r.t. $\theta$. Indeed, for a fixed $\omega$, the selection indices $k_t^{1:N}$ (taking values in a finite set $1:N$) are usually a non-smooth function of the weights $w_t^{1:N}$, which depend on $\theta$.

Therefore the naive FD method using PF cannot be applied in general because of variance explosion of the estimate when $h$ is small, even when using common random number.

## 3.2 Common-indices Finite-Difference method

Let us consider $J_\omega(\theta) = \sum_{t=1}^n r(X_{t,\omega}(\theta))$ making explicit the dependency of the state w.r.t. $\theta$ and a random sample path $\omega$. Under our assumptions, the gradient $\partial J_\omega(\theta)$ is well defined. Now, **let us fix** $\omega$. For clarity, we now omit to write the $\omega$ dependency when no confusion is possible. The function $\theta \mapsto X_t(\theta)$ (for any $1 \le t < n$) is smooth because all transition functions are smooth, the policy is smooth, and the belief state $b_t$ is smooth w.r.t. $\theta$. Underlying the belief feature $b_{t,\theta}(f)$ dependency w.r.t. $\theta$, we write:

$$\theta \stackrel{\text{smooth}}{\longmapsto} b_{t,\theta}(f) \stackrel{\text{smooth}}{\longmapsto} X_t(\theta) \stackrel{\text{smooth}}{\longmapsto} J_\omega(\theta).$$

As already mentioned, the problem with the naive FD method is that the PF estimate $b_{t,\theta}^N(f) = \frac{1}{N} \sum_{i=1}^N f(x_t^i(\theta))$ of $b_{t,\theta}(f)$ is not smooth w.r.t. $\theta$ because it depends on the selection indices $k_{1:t}^{1:N}(\theta)$ which, taken as a function of $\theta$ (through the weights), is not continuous. We write

$$\theta \stackrel{\text{non-smooth}}{\longmapsto} b_{t,\theta}^N(f) = \frac{1}{N} \sum_{i=1}^N f(x_t^i(\theta)) \stackrel{\text{smooth}}{\longmapsto} J_\omega^N(\theta).$$

So a natural idea to recover continuity in a FD method would consists in using exactly the same selection indices for quantities related to $\theta + h$ and $\theta - h$. However, using the same indices means using the same weights during the selection procedure for both trajectories. But this would lead to a wrong estimator because the weights strongly depends on $\theta$ through the observation function $g$.

**Our idea is thus to use the same selection indices but use a likelihood ratio in the belief feature estimation**. More precisely, let us write $k_t^{1:N}(\theta)$ the selection indices obtained for parameter $\theta$, and consider a parameter $\theta'$ in a small neighborhood of $\theta$. Then, an PF estimate for $b_{t,\theta'}(f)$ is

$$b_{t,\theta'}^N(f) \overset{\text{def}}{=} \sum_{i=1}^N \frac{l_t^i(\theta,\theta')}{\sum_{j=1}^N l_t^j(\theta,\theta')} f(x_t^i(\theta')), \text{ with } l_t^i(\theta,\theta') \overset{\text{def}}{=} \frac{\prod_{s=1}^t g(x_s^i(\theta'),y_s(\theta'))}{\prod_{s=1}^t g(x_s^i(\theta),y_s(\theta))} \quad (3)$$

being the likelihood ratios computed along the particle paths, and where the particles $x_{1:t}^{1:N}(\theta')$ have been generated using the same selection indices $k_{1:t}^{1:N}(\theta)$ (and the same random sample path $\omega$) as those used for $\theta$. The next result states the consistency of this estimate and is our main contribution (see (Coquelin et al., 2008) for the proof).

**Proposition 2.** *Under weak conditions on $f$ (see e.g. (Del Moral & Miclo, 2000)), there exists a neighborhood of $\theta$, such that for any $\theta'$ in this neighborhood, $b_{t,\theta'}^N(f)$ defined by (3) is a consistent estimator of $b_{t,\theta'}(f)$, i.e. $\lim_{N \to \infty} b_{t,\theta'}^N(f) = b_{t,\theta'}(f)$ almost surely.*

Thus, for any perturbed value $\theta'$ around $\theta$, we may run an PF where in the resampling step, we use the same selection indices $k_{1:n}^{1:N}(\theta)$ as those obtained for $\theta$. Thus the mapping $\theta' \mapsto b_{t,\theta'}^N(f)$ is smooth. We write:

$$\theta' \overset{\text{smooth}}{\longmapsto} b_{t,\theta'}^N(f) \text{ defined by (3)} \overset{\text{smooth}}{\longmapsto} J_\omega^N(\theta').$$

From the previous proposition we deduce that $J_\omega^N(\theta)$ is a consistent estimator for $J_\omega(\theta)$.

A possible implementation for the gradient estimation is described by Algorithm 3. The algorithm works by updating 3 families of state, observation, and particle populations, denoted by '+', '-', and 'o' for the values of the parameter $\theta + h$, $\theta - h$, and $\theta$ respectively. For the performance measure defined by (1), the algorithm returns the **common indices FD** estimator: $\partial_h J_\omega^N \overset{\text{def}}{=} \frac{1}{2h} \sum_{t=1}^n r(x_t^+) - r(x_t^-)$ where $x_{1:n}^+$ and $x_{1:n}^-$ are upper and lower trajectories simulated under the random sample path $\omega$. Note that although the selection indices are the same, the particle populations '+', '-', and 'o' are different, but very close (when $h$ is small). Hence the likelihood ratios $l_t^{1:N}$ converge to 1 when $h \to 0$, which avoids a source of variance when $h$ is small.

The resulting estimator $\partial_h^M J_\omega^N \overset{\text{def}}{=} \frac{1}{M} \sum_{m=1}^M \partial_h J_{\omega_m}^N$ for $J(\theta)$ would calculate an average over $M$ sample paths $\omega_{1:M}$ of the return of Algorithm 3 called $M$ times. This estimator overcomes the drawbacks of the naive FD estimate: Its **asymptotic bias is of order** $O(h^2)$ (like any centered FD scheme) but **its variance is of order** $O(N^{-1}M^{-1})$ (the Central Limit Theorem applies to the belief feature estimator (3) thus to $\partial_h J_\omega^N$ as well). Since the variance does not degenerate when $h$ is small, one should choose $h$ as small as possible to reduce the mean-squared estimation error.

The complexity of Algorithm 3 is linear in the number of particles $N$. Note that in the current implementation we used 3 populations of particles per derivative. Of course, we could consider a non-centered FD scheme approximating the derivative with $\frac{J(\theta+h)-J(\theta)}{h}$, which is of first order but which only requires 2 particle populations. If the parameter is multidimensional, the full gradient estimate could be obtained by using $K + 1$ populations of particles. Of course, in gradient ascent methods, such FD gradient estimate may be advantageously combined with clever techniques such as simultaneous perturbation stochastic approximation (Spall, 2000), conjugate or second-order gradient approaches.

Note that when $h \to 0$, our estimator converges to an **Infinitesimal Perturbation Analysis** (IPA) estimator (Glasserman, 1991). The same ideas as those presented above could be used to derive an IPA estimator. The advantage of IPA is that it would use one population of particles only (for the full gradient) which may be interesting when the number of parameters $K$ is large. However, the main drawback is that this approach would require to compute analytically the derivatives of all the functions w.r.t. their respective variables, which may be time consuming for the programmer.

## 4 Numerical Experiment

Because of space constraints, our purpose here is simply to illustrate numerically the theoretical findings of previous FD methods (in terms of bias-variance contributions) rather than to provide a full example of POMDP policy optimization. We consider a very simple navigation task for a 2d robot. The robot is defined by its coordinates $x_t \in \mathbb{R}^2$. The observation is a noisy measurement

---

**Algorithm 3** Common-indices Finite Difference estimate of $\partial J_\omega$

---

**Initialize likelihood ratios:**
Set $l_0^{1:N,+} = 1, l_0^{1:N,-} = 1$,
**for** $t = 1$ **to** $n$ **do**
    **State processes:** Sample $u_{t-1} \sim \nu$ and
    Set $x_t^o = F(x_{t-1}^o, a_{t-1}^o, u_{t-1})$, set $x_t^+ = F(x_{t-1}^+, a_{t-1}^+, u_{t-1})$, set $x_t^- = F(x_{t-1}^-, a_{t-1}^-, u_{t-1})$,
    **Observation processes:** Sample $v_t \sim \nu$ and
    Set $y_t^o = G(x_t^o, v_t)$, set $y_t^+ = G(x_t^+, v_t)$, set $y_t^- = G(x_t^-, v_t)$,
    **Particle transition step:** Draw $u_{t-1}^{1:N} \overset{iid}{\sim} \nu$ and
    Set $\widetilde{x}_t^{1:N,o} = F(x_{t-1}^{1:N,o}, a_{t-1}^o, u_{t-1}^{1:N})$,
    Set $\widetilde{x}_t^{1:N,+} = F(x_{t-1}^{1:N,+}, a_{t-1}^+, u_{t-1}^{1:N})$, set $\widetilde{x}_t^{1:N,-} = F(x_{t-1}^{1:N,-}, a_{t-1}^-, u_{t-1}^{1:N})$,
    Set $w_t^{1:N} = \frac{g(\widetilde{x}_t^{1:N,o}, y_t^o)}{\sum_{j=1}^N g(\widetilde{x}_t^{j,o}, y_t^o)}$,
    Set $l_t^{1:N,+} = \frac{g(\widetilde{x}_t^{1:N,+}, y_t^+)}{g(\widetilde{x}_t^{1:N,o}, y_t^o)} l_{t-1}^{1:N,+}$, set $l_t^{1:N,-} = \frac{g(\widetilde{x}_t^{1:N,-}, y_t^-)}{g(\widetilde{x}_t^{1:N,o}, y_t^o)} l_{t-1}^{1:N,-}$,
    **Particle resampling step:**
    Let $k_t^{1:N}$ be the selection indices obtained from the weights $w_t^{1:N}$,
    Set $x_t^{1:N,o} = \widetilde{x}_t^{k_t^{1:N},o}$, set $x_t^{1:N,+} = \widetilde{x}_t^{k_t^{1:N},+}$, set $x_t^{1:N,-} = \widetilde{x}_t^{k_t^{1:N},-}$,
    Set $l_t^{1:N,+} = l_t^{k_t^{1:N},+}$, set $l_t^{1:N,-} = l_t^{k_t^{1:N},-}$,
    **Actions:**
    Set $a_t^o = \pi_\theta\big(\frac{1}{N} \sum_{i=1}^N f(x_t^{i,o})\big)$,
    Set $a_t^+ = \pi_{\theta+h}\big(\sum_{i=1}^N \frac{l_t^{i,+}}{\sum_{j=1}^N l_t^{j,+}} f(x_t^{i,+})\big)$, set $a_t^- = \pi_{\theta-h}\big(\sum_{i=1}^N \frac{l_t^{i,-}}{\sum_{j=1}^N l_t^{j,-}} f(x_t^{i,-})\big)$,
**end for**
**Return:** $\partial_h J_\omega^N \overset{def}{=} \sum_{t=1}^n \frac{r(x_t^+) - r(x_t^-)}{2h}$.

---

of the squared distance to the origin (the goal): $y_t \overset{def}{=} ||x_t||^2 + v_t$, where $v_t \overset{iid}{\sim} \mathcal{N}(0, \sigma_y^2)$ ($\sigma_y^2$ is the variance of the noise). At each time step, the agent may choose a direction $a_t$ (with $||a_t|| = 1$), which results in moving the state, of a step $d$, in the corresponding direction: $x_{t+1} = x_t + da_t + u_t$, where $u_t \overset{i.i.d.}{\sim} \mathcal{N}(0, \sigma_x^2 I)$ is an additive noise. The initial state $x_1$ is drawn from $\nu$, a uniform distribution over the square $[-1, 1]^2$. We consider a class of policies that depend on a single feature belief: the mean of the belief state (i.e. $f(x) = x$). The PF-based policy thus uses the barycenter of the particle population $m_t \overset{def}{=} \frac{1}{N} \sum_{i=1}^N x_t^i$. Let us write $m^\perp$ the $+90^o$ rotation of a vector $m$. We consider policies $\pi_\theta(m) = \frac{-(1-\theta)m + \theta m^\perp}{||-(1-\theta)m + \theta m^\perp||}$ parameterized by $\theta \in [0, 1]$. The chosen action is thus $a_t = \pi_\theta(m_t)$. If the robot was well localized (i.e. $m_t$ close to $x_t$), then the policy $\pi_{\theta=0}$ would move the robot towards the direction of the goal, whereas $\pi_{\theta=1}$ would move it in an orthogonal direction.

The performance measure (to be minimized) is defined as $J(\theta) = \mathbb{E}[||x_n||^2]$, where $n$ is a fixed time. We plot in Figure 2 the performance and gradient estimation obtained when running Algorithms 2 and 3, respectively. We used the numerical values: $N = 10^3$, $M = 10^2$, $h = 10^{-6}$, $n = 10$, $\sigma_x = 0.05$, $\sigma_y = 0.05$, $d = 0.1$.

It is interesting to note that in this problem, the performance is optimal for $\theta^* \simeq 0.3$ (which is slightly better than for $\theta = 0$). $\theta = 0$ would correspond to the best feed-back policy if the state was perfectly known. However, moving in an direction orthogonal to the goal helps improving localization. Here, the optimal policy exhibits a tradeoff between greedy optimization and localization.

| | $h = 10^0$ | $h = 10^{-2}$ | $h = 10^{-4}$ | $h = 10^{-6}$ |
|---|---|---|---|---|
| Bias / Variance NFD | $0.57 / 6.05 \times 10^{-3}$ | $0.31 / 0.13$ | unreliable / $25.3$ | unreliable / $6980$ |
| Bias / Variance CIFD | $0.428 / 0.022$ | $0.00192 / 0.019$ | $0.00247 / 0.02$ | $0.00162 / 0.0188$ |

The table above shows the (empirically measured) bias and variance of the naive FD (NFD) (using common random numbers) method and the common indices FD (CIFD) method, for a specific value $\theta = 0.5$ (with $N = 10^3$, $M = 500$). As predicted, the variance of the NFD approach makes this method inapplicable, whereas that of the CIFD is reasonable.

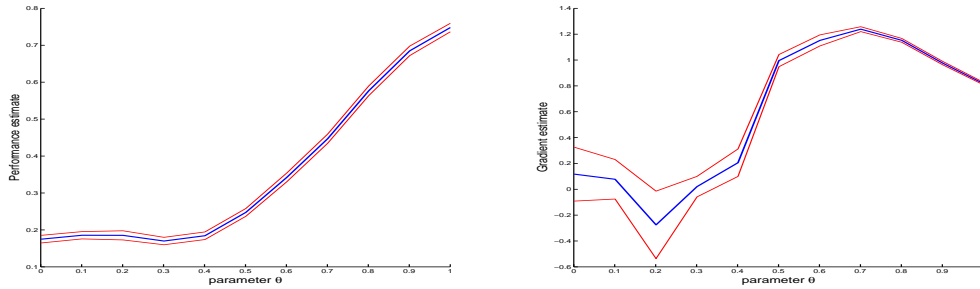

Figure 2: Left: Estimator $\frac{1}{M}\sum_{m=1}^{M} J_{\omega_m}^N(\theta)$ of $J(\theta)$ and confidence intervals $\pm\sqrt{\text{Var}[J_\omega^N(\theta)]/M}$.
Right: Estimator $\frac{1}{M}\sum_{m=1}^{M} \partial_h J_{\omega_m}^N(\theta)$ of $\partial J(\theta)$ and confidence intervals $\pm\sqrt{\text{Var}[\partial_h J_\omega^N(\theta)]/M}$.

## References

Andrieu, C., Doucet, A., Singh, S., & Tadic, V. (2004). Particle methods for change detection, identification and control. *Proceedings of the IEEE*, *92*, 423–438.

Baxter, J., & Bartlett, P. (1999). Direct gradient-based reinforcement learning. *Journal of Artificial Inteligence Reseach*.

Cappé, O., Douc, R., & Moulines, E. (2005). Comparaison of resampling schemes for particle filtering. *4th International Symposium on Image and Signal Processing and Analysis*.

Cérou, F., LeGland, F., & Newton, N. (2001). *Stochastic particle methods for linear tangent filtering equations*, 231–240. IOS Press, Amsterdam.

Coquelin, P., Deguest, R., & Munos, R. (2008). *Sensitivity analysis in particle filters. Application to policy optimization in POMDPs* (Technical Report). INRIA, RR-6710.

Del Moral, P. (2004). *Feynman-kac formulae, genealogical and interacting particle systems with applications*. Springer.

Del Moral, P., & Miclo, L. (2000). Branching and interacting particle systems. approximations of feynman-kac formulae with applications to non-linear filtering. *Séminaire de probabilités de Strasbourg*, *34*, 1–145.

Douc, R., & Moulines, E. (2008). Limit theorems for weighted samples with applications to sequential monte carlo methods. *To appear in Annals of Statistics*.

Doucet, A., Freitas, N. D., & Gordon, N. (2001). *Sequential monte carlo methods in practice*. Springer.

Doucet, A., & Tadic, V. (2003). Parameter estimation in general state-space models using particle methods. *Ann. Inst. Stat. Math*.

Fichoud, J., LeGland, F., & Mevel, L. (2003). *Particle-based methods for parameter estimation and tracking : numerical experiments* (Technical Report 1604). IRISA.

Fox, D., Thrun, S., Burgard, W., & Dellaert, F. (2001). Particle filters for mobile robot localization. *Sequential Monte Carlo Methods in Practice*. New York: Springer.

Glasserman, P. (1991). *Gradient estimation via perturbation analysis*. Kluwer.

Glasserman, P. (2003). *Monte carlo methods in financial engineering*. Springer.

Gordon, N., Salmond, D., & Smith, A. F. M. (1993). Novel approach to nonlinear and non-gaussian bayesian state estimation. *Proceedings IEE-F* (pp. 107–113).

Kaelbling, L. P., Littman, M. L., & Cassandra, A. R. (1998). Planning and acting in partially observable stochastic domains. *Artificial Intelligence*, *101*, 99–134.

Kitagawa, G. (1996). Monte-Carlo filter and smoother for non-Gaussian nonlinear state space models. *J. Comput. Graph. Stat.*, *5*, 1–25.

Lovejoy, W. S. (1991). A survey of algorithmic methods for partially observable Markov decision processes. *Annals of Operations Research*, *28*, 47–66.

Poyadjis, G., Doucet, A., & Singh, S. (2005). Particle methods for optimal filter derivative: Application to parameter estimation. *IEEE ICASSP*.

Rabiner, L. R. (1989). A tutorial on hidden Markov models and selected applications in speech recognition. *Proceedings of the IEEE*, *77*, 257–286.

Spall, J. C. (2000). Adaptive stochastic approximation by the simultaneous perturbation method. *IEEE transaction on automatic control*, *45*, 1839–1853.

